# Toward a Single-Cell Account for Binocular Disparity Tuning: An Energy Model May be Hiding in Your Dendrites

**Bartlett W. Mel**
Department of Biomedical Engineering
University of Southern California, MC 1451
Los Angeles, CA 90089
mel@quake.usc.edu

**Daniel L. Ruderman**
The Salk Institute
10010 N. Torrey Pines Road
La Jolla, CA 92037
ruderman@salk.edu

**Kevin A. Archie**
Neuroscience Program
University of Southern California
Los Angeles, CA 90089
karchie@quake.usc.edu

## Abstract

Hubel and Wiesel (1962) proposed that complex cells in visual cortex are driven by a pool of simple cells with the same preferred orientation but different spatial phases. However, a wide variety of experimental results over the past two decades have challenged the pure hierarchical model, primarily by demonstrating that many complex cells receive monosynaptic input from unoriented LGN cells, or do not depend on simple cell input. We recently showed using a detailed biophysical model that nonlinear interactions among synaptic inputs to an excitable dendritic tree could provide the nonlinear subunit computations that underlie complex cell responses (Mel, Ruderman, & Archie, 1997). This work extends the result to the case of complex cell binocular disparity tuning, by demonstrating in an isolated model pyramidal cell (1) disparity tuning at a resolution much finer than the the overall dimensions of the cell's receptive field, and (2) systematically shifted optimal disparity values for rivalrous pairs of light and dark bars—both in good agreement with published reports (Ohzawa, DeAngelis, & Freeman, 1997). Our results reemphasize the potential importance of intradendritic computation for binocular visual processing in particular, and for cortical neurophysiology in general.

# 1   Introduction

Binocular disparity is a powerful cue for depth in vision. The neurophysiological basis for binocular disparity processing has been of interest for decades, spawned by the early studies of Hubel and Wiesel (1962) showing neurons in primary visual cortex which could be driven by both eyes. Early qualitative models for disparity tuning held that a binocularly driven neuron could represent a particular disparity (zero, near, or far) via a relative shift of receptive field (RF) centers in the right and left eyes. According to this model, a binocular cell fires maximally when an optimal stimulus, e.g. an edge of a particular orientation, is simultaneously centered in the left and right eye receptive fields, corresponding to a stimulus at a specific depth relative to the fixation point. An account of this kind is most relevant to the case of a cortical "simple" cell, whose phase-sensitivity enforces a preference for a particular absolute location and contrast polarity of a stimulus within its monocular receptive fields.

This global receptive field shift account leads to a conceptual puzzle, however, when binocular *complex* cell receptive fields are considered instead, since a complex cell can respond to an oriented feature nearly independent of position within its monocular receptive field. Since complex cell receptive field diameters in the cat lie in the range of 1-3 degrees, the excessive "play" in their monocular receptive fields would seem to render complex cells incapable of signaling disparity on the much finer scale needed for depth perception (measured in minutes).

Intriguingly, various authors have reported that a substantial fraction of complex cells in cat visual cortex are in fact tuned to left-right disparities much finer than that suggested by the size of the monocular RF's. For such cells, a stimulus delivered at the proper disparity, regardless of absolute position in either eye, produces a neural response in excess of that predicted by the sum of the monocular responses (Pettigrew, Nikara, & Bishop, 1968; Ohzawa, DeAngelis, & Freeman, 1990; Ohzawa et al., 1997). Binocular responses of this type suggest that for these cells, the left and right RF's are combined via a correlation operation rather than a simple sum (Nishihara & Poggio, 1984; Koch & Poggio, 1987). This computation has also been formalized in terms of an "energy" model (Ohzawa et al., 1990, 1997), building on the earlier use of energy models to account for complex cell orientation tuning (Pollen & Ronner, 1983) and direction selectivity (Adelson & Bergen, 1985). In an energy model for binocular disparity tuning, sums of linear Gabor filter outputs representing left and right receptive fields are squared to produce the crucial multiplicative cross terms (Ohzawa et al., 1990, 1997).

Our previous biophysical modeling work has shown that the dendritic tree of a cortical pyramidal cells is well suited to support an approximative high-dimensional quadratic input-output relation, where the second-order multiplicative cross terms arise from local interactions among synaptic inputs carried out in quasi-isolated dendritic "subunits" (Mel, 1992b, 1992a, 1993). We recently applied these ideas to show that the position-invariant orientation tuning of a monocular complex cell could be computed within the dendrites of a single cortical cell, based exclusively upon excitatory inputs from a uniform, overlapping population of unoriented ON and OFF cells (Mel et al., 1997). Given the similarity of the "energy" formulations previously proposed to account for orientation tuning and binocular disparity tuning, we hypothesized that a similar type of dendritic subunit computation could underlie disparity tuning in a binocularly driven complex cell.

| Parameter | Value |
|---|---|
| $R_m$ | $10\text{k}\Omega\text{cm}^2$ |
| $R_a$ | $200\Omega\text{cm}$ |
| $C_m$ | $1.0\mu\text{F}/\text{cm}^2$ |
| $V_{\text{rest}}$ | -70 mV |
| Compartments | 615 |
| Somatic $\bar{g}_{\text{Na}}, \bar{g}_{\text{DR}}$ | $0.20, 0.12 \text{ S}/\text{cm}^2$ |
| Dendritic $\bar{g}_{\text{Na}}, \bar{g}_{\text{DR}}$ | $0.05, 0.03 \text{ S}/\text{cm}^2$ |
| Input frequency | $0 - 100$ Hz |
| $\bar{g}_{\text{AMPA}}$ | $0.027$ nS $- 0.295$ nS |
| $\tau_{\text{AMPA}}(on, off)$ | 0.5 ms, 3 ms |
| $\bar{g}_{\text{NMDA}}$ | $0.27$ nS $- 2.95$ nS |
| $\tau_{\text{NMDA}}(on, off)$ | 0.5 ms, 50 ms |
| $E_{\text{syn}}$ | 0 mV |

Table 1: Biophysical simulation parameters. Details of HH channel implementation are given elsewhere (Mel, 1993); original HH channel implementation courtesy Ojvind Bernander and Rodney Douglas. In order that local EPSP size be held approximately constant across the dendritic arbor, peak synaptic conductance at dendritic location $x$ was approximately scaled to the local input resistance (inversely), given by $\bar{g}_{\text{syn}}(x) = c/\tilde{R}_{in}(x)$, where $c$ was a constant, and $\tilde{R}_{in}(x) = \max(R_{in}(x), 200M\Omega)$. Input resistance $R_{in}(x)$ was measured for a passive cell. Thus $\bar{g}_{\text{syn}}$ was identical for all dendritic sites with input resistance below $200M\Omega$, and was given by the larger conductance value shown; roughly 50% of the tree fell within a factor of 2 of this value. Peak conductances at the finest distal tips were smaller by roughly a factor of 10 (smaller number shown). Somatic input resistance was near $24M\Omega$. The peak synaptic conductance values used were such that the ratio of steady state current injection through NMDA vs. AMPA channels was $1.2 \pm 0.4$. Both AMPA and NMDA-type synaptic conductances were modeled using the kinetic scheme of Destexhe et al. (1994); synaptic activation and inactivation time constants are shown for each.

## 2   Methods

Compartmental simulations of a pyramidal cell from cat visual cortex (morphology courtesy of Rodney Douglas and Kevan Martin) were carried out in NEURON (Hines, 1989); simulation parameters are summarized in Table 1. The soma and dendritic membrane contained Hodgkin-Huxley-type (HH) voltage-dependent sodium and potassium channels. Following evidence for higher spike thresholds and decremental propagation in dendrites (Stuart & Sakmann, 1994), HH channel density was set to a uniform, 4-fold lower value in the dendritic membrane relative to that of the cell body. Excitatory synapses from LGN cells included both NMDA and AMPA-type synaptic conductances. Since the cell was considered to be isolated from the cortical network, inhibitory input was not modeled. Cortical cell responses were reported as average spike rate recorded at the cell body over the 500 ms stimulus period, excluding the 50 ms initial transient.

The binocular LGN consisted of two copies of the monocular LGN model used previously (Mel et al., 1997), each consisting of a superimposed pair of 64x64 ON and OFF subfields. LGN cells were modeled as linear, half-rectified center-surround filters with centers 7 pixels in width. We randomly subsampled the left and right LGN arrays by a factor of 16 to yield 1,024 total LGN inputs to the pyramidal cell.

A developmental principle was used to determine the spatial arrangement of these 1,024 synaptic contacts onto the dendritic branches of the cortical cell, as follows. A virtual stimulus ensemble was defined for the cell, consisting of the complete set of single vertical light or dark bars presented binocularly at zero-disparity within the cell's receptive field. Within this ensemble, strong pairwise correlations existed among cells falling into vertically aligned groups of the same (ON or OFF) type, and cells in the vertical column at zero horizontal disparity in the other eye. These binocular cohorts of highly correlated LGN cells were labeled mutual "friends". Progressing through the dendritic tree in depth first order, a randomly chosen LGN cell was assigned to the first dendritic site. A randomly chosen "friend" of hers was assigned to the second site, the third site was assigned to a friend of the site 2 input, etc., until all friends in the available subsample were assigned (4 from each eye, on average). If the friends of the connection at site $i$ were exhausted, a new LGN cell was chosen at random for site $i + 1$. In earlier work, this type of synaptic arrangement was shown to be the outcome of a Hebb-type correlational learning rule, in which random, activity independent formation of synaptic contacts acted to slowly randomize the axo-dendritic interface, shaped by Hebbian stabilization of synaptic contacts based on their short-range correlations with other synapses.

## 3   Results

Model pyramidal cells configured in this way exhibited prominent phase-invariant orientation tuning, the hallmark response property of the visual complex cell. Multiple orientation tuning curves are shown, for example, for a monocular complex cell, giving rise to strong tuning for light and dark bars across the receptive field (fig. 1). The bold curve shows the average of all tuning curves for this cell; the half-width at half max is 25°, in the normal range for complex cells in cat visual cortex (Orban, 1984). When the spatial arrangement of LGN synaptic contacts onto the pyramidal cell dendrites was randomly scrambled, leaving all other model parameters unchanged, orientation tuning was abolished in this cell (right frame), confirming the crucial role of spatially-mediated nonlinear synaptic interactions (average curve from left frame is reproduced for comparison).

Disparity-tuning in an orientation-tuned binocular model cell is shown in fig. 2, compared to data from a complex cell in cat visual cortex (adapted from Ohzawa et al. (1997)). Responses to contrast matched (light-light) and contrast non-matched (light-dark) bar pairs were subtracted to produce these plots. The strong diagonal structure indicates that both the model and real cells responded most vigorously when contrast-matched bars were presented at the same horizontal position in the left and right-eye RF's (i.e. at zero-disparity), whereas peak responses to contrast-non-matched bars occured at symmetric near and far, non-zero disparities.

## 4   Discussion

The response pattern illustrated in fig. 2A is highly similar to the response generated by an analytical binocular energy model for a complex cell (Ohzawa et al., 1997):

$$R_C(X_L, X_R) = \{\exp(-kX_L^2)\cos(2\pi f X_L) + \exp(-kX_R^2)\cos(2\pi f X_R)\}^2 + \{\exp(-kX_L^2)\sin(2\pi f X_L) + \exp(-kX_R^2)\sin(2\pi f X_R)\}^2,$$

$$(1)$$

where $X_L$ and $X_R$ are the horizontal bar positions to the two eyes, $k$ is the factor

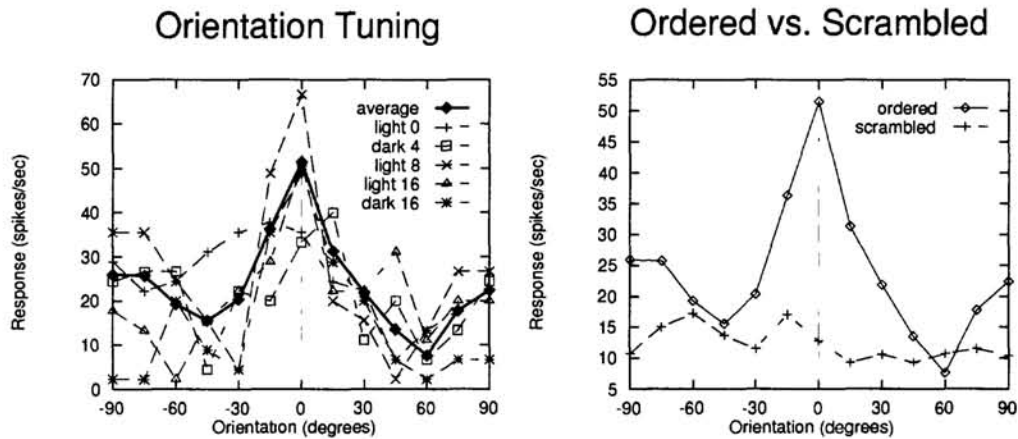

Figure 1: Orientation tuning curves are shown in the left frame for light and dark bars at 3 arbitrary positions. Essentially similar responses were seen at other receptive field positions, and for other complex cells. Bold trace indicates average of tuning curves at positions 0, 1, 2, 4, 8, and 16 for light and dark bars. Similar form of 6 curves shown reflects the translation-invariance of the cell's response to oriented stimuli, and symmetry with respect to ON and OFF input. Orientation tuning is eliminated when the spatial arrangement of LGN synapses onto the model cell dendrites is randomly scrambled (right frame).

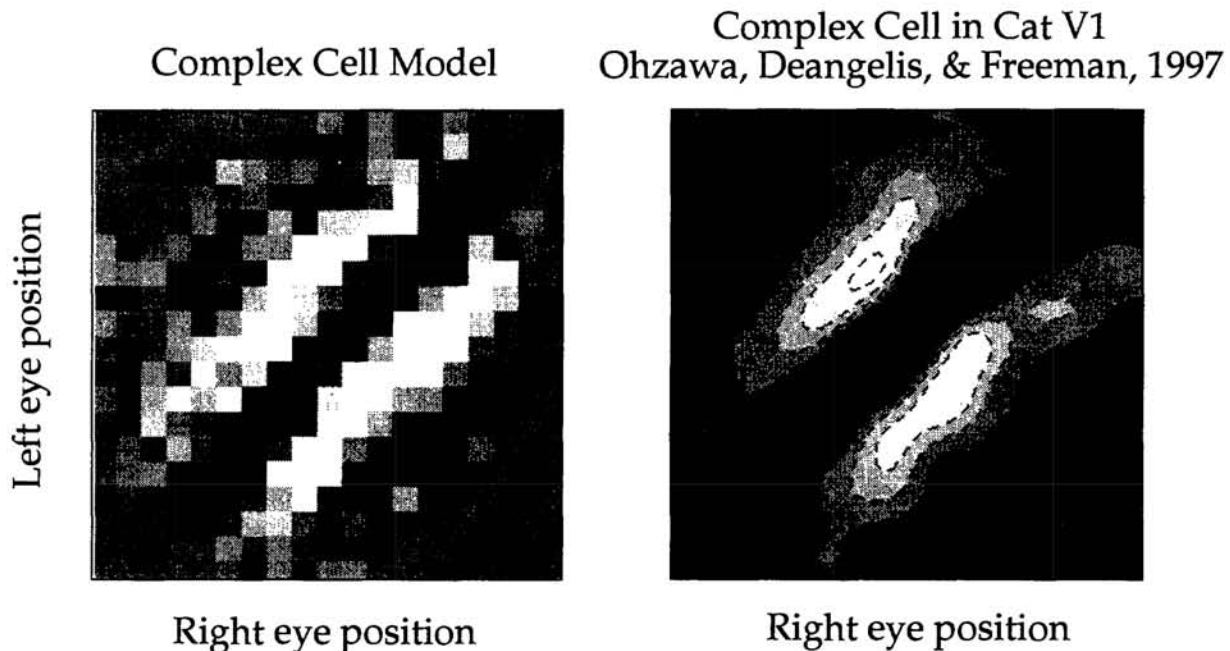

Figure 2: Comparison of disparity tuning in model complex cell to that of a binocular complex cell from cat visual cortex. Light or dark bars were presented simultaneously to the left and right eyes. Bars could be of same polarity in both eyes (light, light) or different polarity (light, dark); cell responses for these two cases were subtracted to produce plot shown in left frame. Right frame shows data similarly displayed for a binocular complex cell in cat visual cortex (adapted from Ohzawa et al. (1997)).

that determines the width of the subunit RF's, and $f$ is the spatial frequency.

In lieu of literal simple cell "subunits", the present results indicate that the subunit computations associated with the terms of an energy model could derive largely from synaptic interactions within the dendrites of the individual cortical cell, driven exclusively by excitatory inputs from unoriented, monocular ON and OFF cells drawn from a uniform overlapping spatial distribution. While lateral inhibition and excitation play numerous important roles in cortical computation, the present results suggest they are not essential for the basic features of the nonlinear disparity tuned responses of cortical complex cells. Further, these results address the paradox as to how inputs from both unoriented LGN cells and oriented simple cells can coexist without conflict within the dendrites of a single complex cell.

A number of controls from previous work suggest that this type of subunit processing is very robustly computed in the dendrites of an individual neuron, with little sensitivity to biophysical parameters and modeling assumptions, including details of the algorithm used to spatially organize the geniculo-cortical projection, specifics of cell morphology, synaptic activation density across the dendritic tree, passive membrane and cytoplasmic parameters, and details of the kinetics, voltage-dependence, or spatial distribution of the voltage-dependent dendritic channels.

One important difference between a standard energy model and the intradendritic responses generated in the present simulation experiments is that the energy model has oriented RF structure at the linear (simple-cell-like) stage, giving rise to oriented, antagonistic ON-OFF subregions (Movshon, Thompson, & Tolhurst, 1978), whereas the linear stage in our model gives rise to center-surround antagonism only within individual LGN receptive fields. Put another way, the LGN-derived subunits in the present model cannot provide all the negative cross-terms that appear in the energy model equations, specifically for pairs of pixels that fall outside the range of a single LGN receptive field.

While the present simulations involve numerous simplifications relative to the full complexity of the cortical microcircuit, the results nonetheless emphasize the potential importance of intradendritic computation in visual cortex.

## Acknowledgements

Thanks to Ken Miller, Allan Dobbins, and Christof Koch for many helpful comments on this work. This work was funded by the National Science Foundation and the Office of Naval Research, and by a Sloan Foundation Fellowship (D.R.).

# References

Adelson, E., & Bergen, J. (1985). Spatiotemporal energy models for the perception of motion. *J. Opt. Soc. Amer.*, A 2, 284–299.

Hines, M. (1989). A program for simulation of nerve equations with branching geometries. *Int. J. Biomed. Comput.*, 24, 55–68.

Hubel, D., & Wiesel, T. (1962). Receptive fields, binocular interaction and functional architecture in the cat's visual cortex. *J. Physiol.*, 160, 106–154.

Koch, C., & Poggio, T. (1987). Biophysics of computation: Neurons, synapses, and membranes. In Edelman, G., Gall, W., & Cowan, W. (Eds.), *Synaptic function*, pp. 637–697. Wiley, New York.

Mel, B. (1992a). The clusteron: Toward a simple abstraction for a complex neuron. In Moody, J., Hanson, S., & Lippmann, R. (Eds.), *Advances in Neural*

*Information Processing Systems, vol. 4*, pp. 35–42. Morgan Kaufmann, San Mateo, CA.

Mel, B. (1992b). NMDA-based pattern discrimination in a modeled cortical neuron. *Neural Computation, 4*, 502–516.

Mel, B. (1993). Synaptic integration in an excitable dendritic tree. *J. Neurophysiol.*, *70*(3), 1086–1101.

Mel, B., Ruderman, D., & Archie, K. (1997). Complex-cell responses derived from center-surround inputs: the surprising power of intradendritic computation. In Mozer, M., Jordan, M., & Petsche, T. (Eds.), *Advances in Neural Information Processing Systems*, Vol. 9, pp. 83–89. MIT Press, Cambridge, MA.

Movshon, J., Thompson, I., & Tolhurst, D. (1978). Receptive field organization of complex cells in the cat's striate cortex. *J. Physiol., 283*, 79–99.

Nishihara, H., & Poggio, T. (1984). Stereo vision for robotics. In Brady, & Paul (Eds.), *Proceedings of the First International Symposium of Robotics Research*, pp. 489–505. MIT Press, Cambridge, MA.

Ohzawa, I., DeAngelis, G., & Freeman, R. (1990). Stereoscopic depth discrimination in the visual cortex: Neurons ideally suited as disparity detectors. *Science, 249*, 1037–1041.

Ohzawa, I., DeAngelis, G., & Freeman, R. (1997). Encoding of binocular disparity by complex cells in the cat's visual cortex. *J. Neurophysiol., June.*

Orban, G. (1984). *Neuronal operations in the visual cortex.* Springer Verlag, New York.

Pettigrew, J., Nikara, T., & Bishop, P. (1968). Responses to moving slits by single units in cat striate cortex. *Exp. Brain Res., 6*, 373–390.

Pollen, D., & Ronner, S. (1983). Visual cortical neurons as localized spatial frequency filters. *IEEE Trans. Sys. Man Cybern., 13*, 907–916.

Stuart, G., & Sakmann, B. (1994). Active propagation of somatic action potentials into neocortical pyramidal cell dendrites. *Nature, 367*, 69–72.